# Online Classification on a Budget

**Koby Crammer**
Computer Sci. & Eng.
Hebrew University
Jerusalem 91904, Israel
kobics@cs.huji.ac.il

**Jaz Kandola**
Royal Holloway,
University of London
Egham, UK
jaz@cs.rhul.ac.uk

**Yoram Singer**
Computer Sci. & Eng.
Hebrew University
Jerusalem 91904, Israel
singer@cs.huji.ac.il

## Abstract

Online algorithms for classification often require vast amounts of memory and computation time when employed in conjunction with kernel functions. In this paper we describe and analyze a simple approach for an *on-the-fly* reduction of the number of past examples used for prediction. Experiments performed with real datasets show that using the proposed algorithmic approach with a *single* epoch is competitive with the support vector machine (SVM) although the latter, being a batch algorithm, accesses each training example multiple times.

## 1 Introduction and Motivation

Kernel-based methods are widely being used for data modeling and prediction because of their conceptual simplicity and outstanding performance on many real-world tasks. The support vector machine (SVM) is a well known algorithm for finding kernel-based linear classifiers with maximal margin [7]. The *kernel trick* can be used to provide an effective method to deal with very high dimensional feature spaces as well as to model complex input phenomena via embedding into inner product spaces. However, despite generalization error being upper bounded by a function of the margin of a linear classifier, it is notoriously difficult to implement such classifiers efficiently. Empirically this often translates into very long training times. A number of alternative algorithms exist for finding a maximal margin hyperplane many of which have been inspired by Rosenblatt's Perceptron algorithm [6] which is an on-line learning algorithm for linear classifiers. The work on SVMs has inspired a number of modifications and enhancements to the original Perceptron algorithm. These incorporate the notion of margin to the learning and prediction processes whilst exhibiting good empirical performance in practice. Examples of such algorithms include the Relaxed Online Maximum Margin Algorithm (ROMMA) [4], the Approximate Maximal Margin Classification Algorithm (ALMA) [2], and the Margin Infused Relaxed Algorithm (MIRA) [1] which can be used in conjunction with kernel functions.

A notable limitation of kernel based methods is their computational complexity since the amount of computer memory that they require to store the so called support patterns grows linearly with the number prediction errors. A number of attempts have been made to speed up the training and testing of SVM's by enforcing a *sparsity* condition. In this paper we devise an online algorithm that is not only sparse but also generalizes well. To achieve this goal our algorithm employs an *insertion* and *deletion* process. Informally, it can be thought of as *revising* the weight vector after each example on which a prediction mistake

has been made. Once such an event occurs the algorithm adds the new erroneous example (the insertion phase), and then immediately searches for past examples that appear to be redundant given the recent addition (the deletion phase). As we describe later, making this adjustment to the algorithm allows us to modify the standard online proof techniques so as to provide a bound on the total number of examples the algorithm keeps.

This paper is organized as follows. In Sec. 2 we formalize the problem setting and provide a brief outline of our method for obtaining a sparse set of support patterns in an online setting. In Sec. 3 we present both theoretical and algorithmic details of our approach and provide a bound on the number of support patterns that constitute the cache. Sec. 4 provides experimental details, evaluated on three real world datasets, to illustrate the performance and merits of our sparse online algorithm. We end the paper with conclusions and ideas for future work.

## 2    Problem Setting and Algorithms

This work focuses on online additive algorithms for classification tasks. In such problems we are typically given a stream of instance-label pairs $(\mathbf{x}_1, y_1), \ldots, (\mathbf{x}_t, y_t), \ldots$. we assume that each instance is a vector $\mathbf{x}_t \in \mathbb{R}^n$ and each label belongs to a finite set $\mathcal{Y}$. In this and the next section we assume that $\mathcal{Y} = \{-1, +1\}$ but relax this assumption in Sec. 4 where we describe experiments with datasets consisting of more than two labels. When dealing with the task of predicting new labels, thresholded linear classifiers of the form $h(\mathbf{x}) = \text{sign}(\mathbf{w} \cdot \mathbf{x})$ are commonly employed. The vector $\mathbf{w}$ is typically represented as a weighted linear combination of the examples, namely $\mathbf{w} = \sum_t \alpha_t y_t \mathbf{x}_t$ where $\alpha_t \geq 0$. The instances for which $\alpha_t > 0$ are referred to as *support patterns*. Under this assumption, the output of the classifier solely depends on inner-products of the form $\mathbf{x} \cdot \mathbf{x}_t$ the use of kernel functions can easily be employed simply by replacing the standard scalar product with a function $K(\cdot, \cdot)$ which satisfies Mercer conditions [7]. The resulting classification rule takes the form $h(\mathbf{x}) = \text{sign}(\mathbf{w} \cdot \mathbf{x}) = \text{sign}(\sum_t \alpha_t y_t K(\mathbf{x}, \mathbf{x}_t))$.

The majority of additive online algorithms for classification, for example the well known Perceptron [6], share a common algorithmic structure. These online algorithms typically work in *rounds*. On the $t$th round, an online algorithm receives an instance $\mathbf{x}_t$, computes the inner-products $s_t = \sum_{i<t} \alpha_i y_i K(\mathbf{x}_i, \mathbf{x}_t)$ and sets the predicted label to be $\text{sign}(s_t)$. The algorithm then receives the correct label $y_t$ and evaluates whether $y_t s_t \leq \beta_t$. The exact value of parameter $\beta_t$ depends on the specific algorithm being used for classification. If the result of this test is negative, the algorithms do not modify $\mathbf{w}_t$ and thus $\alpha_t$ is implicitly set to 0. Otherwise, the algorithms modifies its classification using a predetermined update rule. Informally we can consider this update to be decomposed into three stages. Firstly, the algorithms choose a non-negative value for $\alpha_t$ (again the exact choice of the parameter $\alpha_t$ is algorithm dependent). Secondly, the prediction vector is replaced with a linear combination of the current vector $\mathbf{w}_t$ and the example, $\mathbf{w}_{t+1} = \mathbf{w}_t + \alpha_t y_t \mathbf{x}_t$. In a third, optional stage (see for example [4]), the norm of the newly updated weight vector is scaled, $\mathbf{w}_{t+1} \leftarrow c_t \mathbf{w}_{t+1}$ for some $c_t > 0$. The various online algorithms differ in the way the values of the parameters $\beta_t, \alpha_t$ and $c_t$ are set. A notable example of an online algorithm is the Perceptron algorithm [6] for which we set $\beta_t = 0$, $\alpha_t = 1$ and $c_t = 1$. More recent algorithms such as the Relaxed Online Maximum Margin Algorithm (ROMMA) [4] the Approximate Maximal Margin Classification Algorithm (ALMA) [2] and the Margin Infused Relaxed Algorithm (MIRA) [1] can also be described in this framework although the constants $\beta_t, \alpha_t$ and $c_t$ are not as simple as the ones employed by the Perceptron algorithm.

An important computational consideration needs to be made when employing kernel functions for machine learning tasks. This is because the amount of memory required to store the so called support patterns grows linearly with the number prediction errors. In

**Input:** Tolerance $\beta$.
**Initialize:** Set $\forall t \ \alpha_t = 0 \ , \mathbf{w}_0 = 0 \ , C_0 = \emptyset$.
**Loop:** For $t = 1, 2, \ldots, T$

- Get a new instance $\mathbf{x}_t \in \mathbb{R}^n$.
- Predict $\hat{y}_t = \text{sign}\left(y_t(\mathbf{x}_t \cdot \mathbf{w}_{t-1})\right)$.
- Get a new label $y_t$.
- if $y_t(\mathbf{x}_t \cdot \mathbf{w}_{t-1}) \leq \beta$ update:
    1. Insert $C_t \leftarrow C_{t-1} \cup \{t\}$.
    2. Set $\alpha_t = 1$.
    3. Compute $\mathbf{w}_t \leftarrow \mathbf{w}_{t-1} + y_t \alpha_t \mathbf{x}_t$.
    4. `DistillCache`$(C_t, \mathbf{w}_t, (\alpha_1, \ldots, \alpha_t))$.

**Output :** $H(\mathbf{x}) = \text{sign}(\mathbf{w}_T \cdot \mathbf{x})$.

Figure 1: The aggressive Perceptron algorithm with a variable-size cache.

this paper we shift the focus to the problem of devising online algorithms which are budget-conscious as they attempt to keep the number of support patterns small. The approach is attractive for at least two reasons. Firstly, both the training time and classification time can be reduced significantly if we store only a fraction of the potential support patterns. Secondly, a classier with a small number of support patterns is intuitively "simpler", and hence are likely to exhibit good generalization properties rather than complex classifiers with large numbers of support patterns. (See for instance [7] for formal results connecting the number of support patterns to the generalization error.)

In Sec. 3 we present a formal analysis and the algorithmic details of our approach. Let us now provide a general overview of how to restrict the number of support patterns in an online setting. Denote by $C_t$ the indices of patterns which constitute the classification vector $\mathbf{w}_t$. That is, $i \in C_t$ if and only if $\alpha_i > 0$ on round $t$ when $\mathbf{x}_t$ is received. The online classification algorithms discussed above keep enlarging $C_t$ – once an example is added to $C_t$ it will never be deleted. However, as the online algorithm receives more examples, the performance of the classifier improves, and some of the past examples may have become redundant and hence

**Input:** $C, \mathbf{w}, (\alpha_1, \ldots, \alpha_t)$.
**Loop:**

- Choose $i \in C$ such that $\beta \leq y_i(\mathbf{w} - \alpha_i y_i \mathbf{x}_i)$.
- if no such $i$ exists then return.
- Remove the example $i$ :
    1. $\alpha_i = 0$.
    2. $\mathbf{w} \leftarrow \mathbf{w} - \alpha_i y_i \mathbf{x}_i$.
    3. $C \leftarrow C/\{i\}$

**Return :** $C, \mathbf{w}, (\alpha_1, \ldots, \alpha_t)$.

Figure 2: DistillCache

can be removed. Put another way, old examples may have been inserted into the cache simply due the lack of support patterns in early rounds. As more examples are observed, the old examples maybe replaced with new examples whose location is closer to the decision boundary induced by the online classifier. We thus add a new stage to the online algorithm in which we discard a few old examples from the cache $C_t$. We suggest a modification of the online algorithm structure as follows. Whenever $y_t\left(\sum_{i<t} \alpha_i y_i K(\mathbf{x}, \mathbf{x}_i)\right) \leq \beta_t$, then after adding $\mathbf{x}_t$ to $\mathbf{w}$ and inserting the $t$th into $C_t$, we scan the cache $C_t$ for seemingly redundant examples by examining the margin conditions of old examples in $C_t$. If such an example is found, we discard it from the both the classifier and the cache by updating $\mathbf{w}_t \leftarrow \mathbf{w}_t - \alpha_i y_i \mathbf{x}_i$ and setting $C_t \leftarrow C_t/\{i\}$. The pseudocode for this "budget-conscious" version of the aggressive Perceptron algorithm [3] is given in Fig. 1. We say that the algo-

rithm employs a *variable-size* cache since we do no limit explicitly the number of support patterns though we do attempt to discard as many patterns as possible from the cache. A similar modification, to that described for aggressive Perceptron, can be made to all of the online classification algorithms outlined above. In particular, we use a modification of the MIRA [1] algorithm in our experiments.

## 3 Analysis

In this section we provide our main formal result for the algorithm described in the previous section. Informally, the theorem below states that the actual size of the cache that the algorithm builds is inversely proportional to the square of the best margin that can be achieved on the data. This form of bound is common to numerous online learning algorithms for classification. However, here the bound is on the *size* of the cache whereas in common settings the corresponding bounds are on the number of prediction *mistakes*. The bound also depends on $\beta$, the margin used by the algorithm to check whether a new example should be added to the cache and to discard old examples attaining a large margin. Clearly, the larger the value of $\beta$ the more often we add examples to the cache.

**Theorem 1** *Let* $(\mathbf{x}_1, y_1), \ldots, (\mathbf{x}_T, y_T)$ *be an input sequence for the algorithm given in Fig. 1, where* $\mathbf{x}_t \in \mathbb{R}^n$ *and* $y_t \in \{-1, +1\}$. *Denote by* $R = \max_t \|\mathbf{x}_t\|$. *Assume that there exists a vector* $\mathbf{u}$ *of unit norm* $(\|\mathbf{u}\| = 1)$ *which classifies the entire sequence correctly with a margin* $\gamma = \min_t y_t(\mathbf{u} \cdot \mathbf{x}_t) > 0$. *Then the number of support patterns constituting the cache is at most* $S \le (R^2 + 2\beta)/\gamma^2$ .

**Proof:** The proof of the theorem is based on the mistake bound of the Perceptron algorithm [5]. To prove the theorem we bound $\|\mathbf{w}_T\|_2^2$ from above and below and compare the bounds. Denote by $\alpha_i^t$ the weight of the $i$th example at the end of round $t$ (*after* stage 4 of the algorithm). Similarly, we denote by $\tilde{\alpha}_i^t$ to be the weight of the $i$th example on round $t$ after stage 3, before calling the DistillCache (Fig. 2) procedure. We analogously denote by $\mathbf{w}_t$ and $\tilde{\mathbf{w}}_t$ the corresponding instantaneous classifiers. First, we derive a lower bound on $\|\mathbf{w}_T\|^2$ by bounding the term $\mathbf{w}_T \cdot \mathbf{u}$ from below in a recursive manner.

$$
\begin{aligned}
\mathbf{w}_T \cdot \mathbf{u} &= \sum_{t \in C_T} \alpha_t^T y_t(\mathbf{x}_t \cdot \mathbf{u}) \\
&\ge \gamma \sum_{t \in C_T} \alpha_t^T = \gamma S .
\end{aligned}
\tag{1}
$$

We now turn to upper bound $\|\mathbf{w}_T\|^2$. Recall that each example may be added to the cache and removed from the cache a single time. Let us write $\|\mathbf{w}_T\|^2$ as a telescopic sum,

$$
\|\mathbf{w}_T\|^2 = (\|\mathbf{w}_T\|^2 - \|\tilde{\mathbf{w}}_T\|^2) + (\|\tilde{\mathbf{w}}_T\|^2 - \|\mathbf{w}_{T-1}\|^2) + \ldots + (\|\tilde{\mathbf{w}}_1\|^2 - \|\mathbf{w}_0\|^2) . \tag{2}
$$

We now consider three different scenarios that may occur for each new example. The first case is when we did not insert the $t$th example into the cache at all. In this case, $(\|\tilde{\mathbf{w}}_t\|^2 - \|\mathbf{w}_{t-1}\|^2) = 0$. The second scenario is when an example is inserted into the cache and is never discarded in future rounds, thus,

$$
\|\tilde{\mathbf{w}}_t\|^2 = \|\mathbf{w}_{t-1} + y_t \mathbf{x}_t\|^2 = \|\mathbf{w}_{t-1}\|^2 + 2y_t(\mathbf{w}_{t-1} \cdot \mathbf{x}_t) + \|\mathbf{x}_t\|^2 .
$$

Since we inserted $(\mathbf{x}_t, y_t)$, the condition $y_t(\mathbf{w}_{t-1} \cdot \mathbf{x}_t) \le \beta$ must hold. Combining this with the assumption that the examples are enclosed in a ball of radius $R$ we get, $(\|\tilde{\mathbf{w}}_t\|^2 - \|\mathbf{w}_{t-1}\|^2) \le 2\beta + R^2$. The last scenario occurs when an example is inserted into the cache on some round $t$, and is then later on removed from the cache on round $t + p$ for $p > 0$. As in the previous case we can bound the value of summands in Equ. (2),

$$
(\|\tilde{\mathbf{w}}_t\|^2 - \|\mathbf{w}_{t-1}\|^2) + (\|\mathbf{w}_{t+p}\|^2 - \|\tilde{\mathbf{w}}_{t+p}\|^2)
$$

**Input:** Tolerance $\beta$, Cache Limit $n$.
**Initialize:** Set $\forall t \;\; \alpha_t = 0 \;\;, \mathbf{w}_0 = 0 \;\;, C_0 = \emptyset$.
**Loop:** For $t = 1, 2, \ldots, T$

- Get a new instance $\mathbf{x}_t \in \mathbb{R}^n$.
- Predict $\hat{y}_t = \text{sign}\left(y_t(\mathbf{x}_t \cdot \mathbf{w}_{t-1})\right)$.
- Get a new label $y_t$.
- if $y_t(\mathbf{x}_t \cdot \mathbf{w}_{t-1}) \leq \beta$ update:
    1. If $|C_t| = n$ remove one example:
        (a) Find $i = \arg\max_{j \in C_t}\{y_j(\mathbf{w}_{t-1} - \alpha_j y_j \mathbf{x}_j)\}$.
        (b) Update $\mathbf{w}_{t-1} \leftarrow \mathbf{w}_{t-1} - \alpha_i y_i \mathbf{x}_i$.
        (c) Remove $C_{t-1} \leftarrow C_{t-1}/\{i\}$
    2. Insert $C_t \leftarrow C_{t-1} \cup \{t\}$.
    3. Set $\alpha_t = 1$.
    4. Compute $\mathbf{w}_t \leftarrow \mathbf{w}_{t-1} + y_t \alpha_t \mathbf{x}_t$.

**Output :** $H(\mathbf{x}) = \text{sign}(\mathbf{w}_T \cdot \mathbf{x})$.

---

Figure 3: The aggressive Perceptron algorithm with as fixed-size cache.

$$
\begin{aligned}
&= \; 2y_t(\mathbf{w}_{t-1} \cdot \mathbf{x}_t) + \|\mathbf{x}_t\|^2 - 2y_t(\tilde{\mathbf{w}}_{t+p} \cdot \mathbf{x}_t) + \|\mathbf{x}_t\|^2 \\
&= \; 2\left[y_t(\mathbf{w}_{t-1} \cdot \mathbf{x}_t) - y_t\left((\tilde{\mathbf{w}}_{t+p} - y_t\mathbf{x}_t) \cdot \mathbf{x}_t\right)\right] \\
&\leq \; 2\left[\beta - y_t\left((\tilde{\mathbf{w}}_{t+p} - y_t\mathbf{x}_t) \cdot \mathbf{x}_t\right)\right] \; .
\end{aligned}
$$

Based on the form of the cache update we know that $y_t\left((\tilde{\mathbf{w}}_{t+p} - y_t\mathbf{x}_t) \cdot \mathbf{x}_t\right) \geq \beta$, and thus,

$$
\left(\|\tilde{\mathbf{w}}_t\|^2 - \|\mathbf{w}_{t-1}\|^2\right) + \left(\|\mathbf{w}_{t+p}\|^2 - \|\tilde{\mathbf{w}}_{t+p}\|^2\right) \leq 0 \; .
$$

Summarizing all three cases we see that only the examples which persist in the cache contribute a factor of $R^2 + 2\beta$ each to the bound of the telescopic sum of Equ. (2) and the rest of the examples do contribute anything to the bound. Hence, we can bound the norm of $\mathbf{w}_T$ as follows,

$$
\|\mathbf{w}_T\|^2 \leq S\left(R^2 + 2\beta\right) \; . \tag{3}
$$

We finish up the proof by applying the Cauchy-Swartz inequality and the assumption $\|\mathbf{u}\| = 1$. Combining Equ. (1) and Equ. (3) we get,

$$
\gamma^2 S^2 \leq (\mathbf{w}_T \cdot \mathbf{u})^2 \leq \|\mathbf{w}_T\|^2 \|\mathbf{u}\|^2 \leq S(2\beta + R^2) \; ,
$$

which gives the desired bound. ∎

## 4  Experiments

In this section we describe the experimental methods that were used to compare the performance of standard online algorithms with the new algorithm described above. We also describe shortly another variant that sets a hard limit on the number of support patterns. The experiments were designed with the aim of trying to answer the following questions. First, what is effect of the number of support patterns on the generalization error (measured in terms of classification accuracy on unseen data), and second, would the algorithm described in Fig. 2 be able to find an optimal cache size that is able to achieve the best generalization performance. To examine each question separately we used a modified version of the algorithm described by Fig. 2 in which we restricted ourselves to have a fixed bounded cache. This modified algorithm (which we refer to as the *fixed budget* Perceptron)

| Name | No. of Training Examples | No. of Test Examples | No. of Classes | No. of Attributes |
|------|------|------|------|------|
| mnist | 60000 | 10000 | 10 | 784 |
| letter | 16000 | 4000 | 26 | 16 |
| usps | 7291 | 2007 | 10 | 256 |

Table 1: Description of the datasets used in experiments.

simulates the original Perceptron algorithm with one notable difference. When the number of support patterns exceeds a pre-determined limit, it chooses a support pattern from the cache and discards it. With this modification the number of support patterns can never exceed the pre-determined limit. This modified algorithm is described in Fig. 3. The algorithm deletes the example which seemingly attains the highest margin after the removal of the example itself (line 1(a) in Fig. 3).

Despite the simplicity of the original Perceptron algorithm [6] its good generalization performance on many datasets is remarkable. During the last few year a number of other additive online algorithms have been developed [4, 2, 1] that have shown better performance on a number of tasks. In this paper, we have preferred to embed these ideas into another online algorithm and start with a higher baseline performance. We have chosen to use the Margin Infused Relaxed Algorithm (MIRA) as our baseline algorithm since it has exhibited good generalization performance in previous experiments [1] and has the additional advantage that it is designed to solve multiclass classification problem directly without any recourse to performing reductions.

The algorithms were evaluated on three natural datasets: mnist[1], usps[2] and letter[3]. The characteristics of these datasets has been summarized in Table 1. A comprehensive overview of the performance of various algorithms on these datasets can be found in a recent paper [2]. Since all of the algorithms that we have evaluated are online, it is not implausible for the specific ordering of examples to affect the generalization performance. We thus report results averaged over 11 random permutations for usps and letter and over 5 random permutations for mnist. No free parameter optimization was carried out and instead we simply used the values reported in [1]. More specifically, the margin parameter was set to $\beta = 0.01$ for all algorithms and for all datasets. A homogeneous polynomial kernel of degree 9 was used when training on the mnist and usps data sets, and a RBF kernel for letter data set. (The variance of the RBF kernel was identical to the one used in [1].)

We evaluated the performance of four algorithms in total. The first algorithm was the standard MIRA online algorithm, which does not incorporate any budget constraints. The second algorithm is the version of MIRA described in Fig. 3 which uses a fixed limited budget. Here we enumerated the cache size limit in each experiment we performed. The different sizes that we tested are dataset dependent but for each dataset we evaluated at least 10 different sizes. We would like to note that such an enumeration cannot be done in an online fashion and the goal of employing the the algorithm with a fixed-size cache is to underscore the merit of the truly adaptive algorithm. The third algorithm is the version of MIRA described in Fig. 2 that adapts the cache size during the running of the algorithms. We also report additional results for a multiclass version of the SVM [1]. Whilst this algorithm is not online and during the training process it considers all the examples at once, this algorithm serves as our gold-standard algorithm against which we want to compare

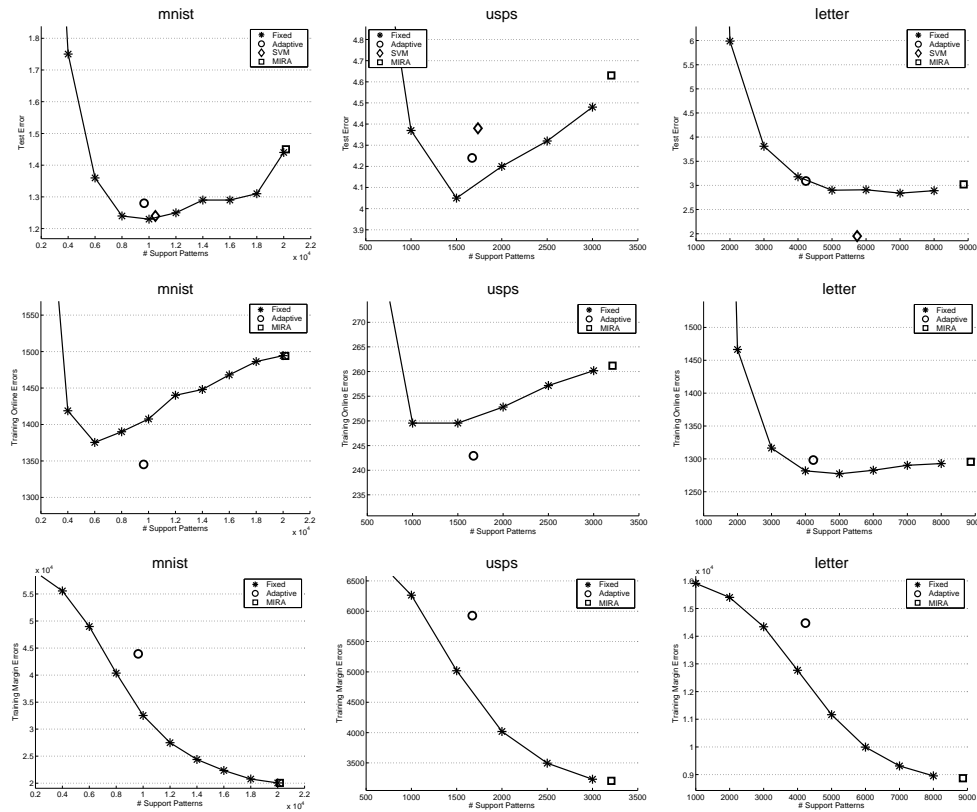

Figure 4: Results on a three data sets - mnist (left), usps (center) and letter (right). Each point in a plot designates the test error (y-axis) vs. the number of support patterns used (x-axis). Four algorithms are compared - SVM, MIRA, MIRA with a fixed cache size and MIRA with a variable cache size.

performance. Note that for the multiclass SVM we report the results using the best set of parameters, which does not coincide with the set of parameters used for the online training.

The results are summarized in Fig 4. This figure is composed of three different plots organized in columns. Each of these plots corresponds to a different dataset - mnist (left), usps (center) and letter (right). In each of the three plots the $x$-axis designates number of support patterns the algorithm uses. The results for the fixed-size cache are connected with a line to emphasize the performance dependency on the size of the cache.

The top row of the three columns shows the generalization error. Thus the $y$-axis designates the test error of an algorithm on unseen data at the end of the training. Looking at the error of the algorithm with a fixed-size cache reveals that there is a broad range of cache size where the algorithm exhibits good performance. In fact for MNIST and USPS there are sizes for which the test error of the algorithm is better than SVM's test error. Naturally, we cannot fix the correct size in hindsight so the question is whether the algorithm with variable cache size is a viable automatic size-selection method. Analyzing each of the datasets in turn reveals that this is indeed the case – the algorithm obtains a very similar number of support patterns and test error when compared to the SVM method. The results are somewhat less impressive for the letter dataset which contains less examples per class. One possible explanation is that the algorithm had fewer chances to modify and distill the cache. Nonetheless, overall the results are remarkable given that all the online algorithms make a single pass through the data and the variable-size method finds a very good cache size while

making it also comparable to the SVM in terms of performance. The MIRA algorithm, which does not incorporate any form of example insertion or deletion in its algorithmic structure, obtains the poorest level of performance not only in terms of generalization error but also in terms of number of support patterns.

The plot of *online* training error against the number of support patterns, in row 2 of Fig 4, can be considered to be a good on-the-fly validation of generalization performance. As the plots indicate, for the fixed and adaptive versions of the algorithm, on all the datasets, a low online training error translates into good generalization performance. Comparing the test error plots with the online error plots we see a nice similarity between the qualitative behavior of the two errors. Hence, one can use the online error, which is easy to evaluate, to choose a good cache size for the fixed-size algorithm.

The third row gives the online training margin errors that translates directly to the number of insertions into the cache. Here we see that the good test error and compactness of the algorithm with a variable cache size come with a price. Namely, the algorithm makes significantly more insertions into the cache than the fixed size version of the algorithm. However, as the upper two sets of plots indicate, the surplus in insertions is later taken care of by excess deletions and the end result is very good overall performance. In summary, the online algorithm with a variable cache and SVM obtains similar levels of generalization and also number of support patterns. While the SVM is still somewhat better in both aspects for the letter dataset, the online algorithm is much simpler to implement and performs a *single* sweep through the training data.

## 5   Summary

We have described and analyzed a new sparse online algorithm that attempts to deal with the computational problems implicit in classification algorithms such as the SVM. The proposed method was empirically tested and its performance in both the size of the resulting classifier and its error rate are comparable to SVM. There are a few possible extensions and enhancements. We are currently looking at alternative criteria for the deletions of examples from the cache. For instance, the weight of examples might relay information on their importance for accurate classification. Incorporating prior knowledge to the insertion and deletion scheme might also prove important. We hope that such enhancements would make the proposed approach a viable alternative to SVM and other batch algorithms.

**Acknowledgements:** The authors would like to thank John Shawe-Taylor for many helpful comments and discussions. This research was partially funded by the EU project KerMIT No. IST-2000-25341.

## Footnotes

[1]Available from http://www.research.att.com/~yann

[2]Available from ftp.kyb.tuebingen.mpg.de

[3]Available from http://www.ics.uci.edu/~mlearn/MLRepository.html

## References

[1] K. Crammer and Y. Singer. Ultraconservative online algorithms for multiclass problems. *Jornal of Machine Learning Research*, 3:951–991, 2003.

[2] C. Gentile. A new approximate maximal margin classification algorithm. *Journal of Machine Learning Research*, 2:213–242, 2001.

[3] Mézard M. Krauth W. Learning algorithms with optimal stability in neural networks. *Journal of Physics A.*, 20:745, 1987.

[4] Y. Li and P. M. Long. The relaxed online maximum margin algorithm. *Machine Learning*, 46(1–3):361–387, 2002.

[5] A. B. J. Novikoff. On convergence proofs on perceptrons. In *Proceedings of the Symposium on the Mathematical Theory of Automata*, volume XII, pages 615–622, 1962.

[6] F. Rosenblatt. The perceptron: A probabilistic model for information storage and organization in the brain. *Psychological Review*, 65:386–407, 1958. (Reprinted in *Neurocomputing* (MIT Press, 1988).).

[7] V. N. Vapnik. *Statistical Learning Theory*. Wiley, 1998.